# Noisy Neural Networks and Generalizations

**Hava T. Siegelmann**
Industrial Eng. and Management, Mathematics
Technion - IIT
Haifa 32000, Israel
*iehava@ie.technion.ac.il*

**Alexander Roitershtein**
Mathematics
Technion - IIT
Haifa 32000, Israel
*roiterst@math.technion.ac.il*

**Asa Ben-Hur**
Industrial Eng. and Management
Technion - IIT
Haifa 32000, Israel
*asa@tx.technion.ac.il*

## Abstract

In this paper we define a probabilistic computational model which generalizes many noisy neural network models, including the recent work of Maass and Sontag [5]. We identify weak ergodicity as the mechanism responsible for restriction of the computational power of probabilistic models to *definite languages*, independent of the characteristics of the noise: whether it is discrete or analog, or if it depends on the input or not, and independent of whether the variables are discrete or continuous. We give examples of weakly ergodic models including noisy computational systems with noise depending on the current state and inputs, aggregate models, and computational systems which update in continuous time.

## 1  Introduction

Noisy neural networks were recently examined, e.g. in. [1, 4, 5]. It was shown in [5] that Gaussian-like noise reduces the power of analog recurrent neural networks to the class of definite languages, which are a strict subset of regular languages. Let $\Sigma$ be an arbitrary alphabet. $L \subseteq \Sigma^*$ is called a *definite language* if for some integer $r$ any two words coinciding on the last $r$ symbols are either both in $L$ or neither in $L$. The ability of a computational system to recognize only definite languages can be interpreted as saying that the system forgets all its input signals, except for the most recent ones. This property is reminiscent of human short term memory.

"Definite probabilistic computational models" have their roots in Rabin's pioneering work on probabilistic automata [9]. He identified a condition on probabilistic automata with a finite state space which restricts them to definite languages. Paz [8] generalized Rabin's condition, applying it to automata with a countable state space, and calling it *weak ergodicity* [7, 8]. In their ground-breaking paper [5],

Maass and Sontag extended the principle leading to definite languages to a finite interconnection of *continuous-valued* neurons. They proved that in the presence of "analog noise" (e.g. Gaussian), recurrent neural networks are limited in their computational power to definite languages. Under a different noise model, Maass and Orponen [4] and Casey [1] showed that such neural networks are reduced in their power to regular languages.

In this paper we generalize the condition of weak ergodicity, making it applicable to numerous probabilistic computational machines. In our general probabilistic model, the state space can be *arbitrary*: it is not constrained to be a finite or infinite set, to be a discrete or non-discrete subset of some Euclidean space, or even to be a metric or topological space. The input alphabet is arbitrary as well (e.g., bits, rationals, reals, etc.). The stochasticity is not necessarily defined via a transition probability function (TPF) as in all the aforementioned probabilistic and noisy models, but through the more general *Markov operators acting on measures*. Our *Markov Computational Systems* (MCS's) include as special cases Rabin's actual probabilistic automata with cut-point [9], the quasi-definite automata by Paz [8], and the noisy analog neural network by Maass and Sontag [5]. Interestingly, our model also includes: analog dynamical systems and neural models, which have no underlying deterministic rule but rather update probabilistically by using finite memory; neural networks with an unbounded number of components; networks of variable dimension (e.g., "recruiting networks"); hybrid systems that combine discrete and continuous variables; stochastic cellular automata; and stochastic coupled map lattices.

We prove that all weakly ergodic Markov systems are stable, i.e. are robust with respect to architectural imprecisions and environmental noise. This property is desirable for both biological and artificial neural networks. This robustness was known up to now only for the classical discrete probabilistic automata [8, 9]. To enable practicality and ease in deciding weak ergodicity for given systems, we provide two conditions on the transition probability functions under which the associated computational system becomes weakly ergodic. One condition is based on a version of Doeblin's condition [5] while the second is motivated by the theory of scrambling matrices [7, 8]. In addition we construct various examples of weakly ergodic systems which include synchronous or asynchronous computational systems, and hybrid continuous and discrete time systems.

## 2    Markov Computational System (MCS)

Instead of describing various types of noisy neural network models or stochastic dynamical systems we define a general abstract probabilistic model. When dealing with systems containing inherent elements of uncertainty (e.g., noise) we abandon the study of individual trajectories in favor of an examination of the flow of state distributions. The noise models we consider are homogeneous in time, in that they may depend on the input, but do not depend on time. The dynamics we consider is defined by operators acting in the space of measures, and are called *Markov operators* [6]. In the following we define the concepts which are required for such an approach.

Let $\Sigma$ be an arbitrary alphabet and $\Omega$ be an abstract state space. We assume that a $\sigma$-algebra $\mathcal{B}$ (not necessarily Borel sets) of subsets of $\Omega$ is given, thus $(\Omega, \mathcal{B})$ is a measurable space. Let us denote by $\mathcal{P}$ the set of probability measures on $(\Omega, \mathcal{B})$. This set is called a *distribution space*.

Let $\mathcal{E}$ be a space of finite measures on $(\Omega, \mathcal{B})$ with the total variation norm defined

by

$$\|\mu\|_1 = |\mu|(\Omega) = \sup_{A \in \mathcal{B}} \mu(A) - \inf_{A \in \mathcal{B}} \mu(A). \tag{1}$$

Denote by $\mathcal{L}$ the set of all bounded linear operators acting from $\mathcal{E}$ to itself. The $\|\cdot\|_1$- norm on $\mathcal{E}$ induces a norm $\|P\|_1 = \sup_{\mu \in \mathcal{P}} \|P\mu\|_1$ in $\mathcal{L}$. An operator $P \in \mathcal{L}$ is said to be a *Markov operator* if for any probability measure $\mu \in \mathcal{P}$, the image $P\mu$ is again a probability measure. For a Markov operator, $\|P\|_1 = 1$.

**Definition 2.1** A *Markov system* is a set of Markov operators $T = \{P_u : u \in \Sigma\}$.

With any Markov system $T$, one can associate a probabilistic computational system. If the probability distribution on the initial states is given by the probability measure $\mu_0$, then the distribution of states after $n$ computational steps on inputs $w = w_0, w_1, ..., w_n$, is defined as in [5, 8]

$$P_w \mu_0(A) = P_{w_n} \cdot \ldots \cdot P_{w_1} P_{w_0} \mu_0. \tag{2}$$

Let $\mathcal{A}$ and $\mathcal{R}$ be two subset of $\mathcal{P}$ with the property of having a $\rho$-gap

$$dist(\mathcal{A}, \mathcal{R}) = \inf_{\mu \in \mathcal{A}, \nu \in \mathcal{R}} \|\mu - \nu\|_1 = \rho > 0 \tag{3}$$

The first set is called a set of *accepting distributions* and the second is called a set of *rejecting distributions*. A language $L \in \Sigma^*$ is said to be *recognized* by Markov computational system $\mathcal{M} = \langle \mathcal{E}, \mathcal{A}, \mathcal{R}, \Sigma, \mu_0, T \rangle$ if

$$w \in L \Leftrightarrow P_w \mu_0 \in \mathcal{A}$$
$$w \notin L, \Leftrightarrow P_w \mu_0 \in \mathcal{R}.$$

This model of language recognition with a gap between accepting and rejecting spaces agrees with Rabin's model of probabilistic automata with isolated cut-point [9] and the model of analog probabilistic computation [4, 5].

An example of a Markov system is a system of operators defined by TPF on $(\Omega, \mathcal{B})$. Let $P_u(x, A)$ be the probability of moving from a state $x$ to the set of states $A$ upon receiving the input signal $u \in \Sigma$. The function $P_u(x, \cdot)$ is a probability measure for all $x \in \Omega$ and $P_u(\cdot, A)$ is a measurable function of $x$ for any $A \in \mathcal{B}$. In this case, $P_u \mu(A)$ are defined by

$$P_u \mu(A) = \int_\Omega P_u(x, A) \mu(dx). \tag{4}$$

## 3  Weakly Ergodic MCS

Let $P \in \mathcal{L}$ be a Markov operator. The real number $\gamma(P) = 1 - \frac{1}{2} \sup_{\mu, \nu \in \mathcal{P}} \|P\mu - P\nu\|_1$ is called *the ergodicity coefficient of the Markov operator*. We denote $\delta(P) = 1 - \gamma(P)$. It can be proven that for any two Markov operators $P_1, P_2$, $\delta(P_1 P_2) \leq \delta(P_1)\delta(P_2)$. The ergodicity coefficient was introduced by Dobrushin [2] for the particular case of Markov operators induced by TPF $P(x, A)$. In this special case $\gamma(P) = 1 - \sup_{x,y} \sup_A |P(x, A) - P(y, A)|$.

Weakly ergodic systems were introduced and studied by Paz in the particular case of a denumerable state space $\Omega$, where Markov operators are represented by infinite dimensional matrices. The following definition makes no assumption on the associated measurable space.

**Definition 3.1** A Markov system $\{P_u, \ u \in \Sigma\}$ is called *weakly ergodic* if for any $\alpha > 0$, there is an integer $r = r(\alpha)$ such that for any $w \in \Sigma^{\geq r}$ and any $\mu, \nu \in \mathcal{P}$,

$$\delta(P_w) = \frac{1}{2}\|P_w\mu - P_w\nu\|_1 \leq \alpha. \tag{5}$$

An MCS $\mathcal{M}$ is called *weakly ergodic* if its associated Markov system $\{P_u, \ u \in \Sigma\}$ is weakly ergodic. ∎

An MCS $\mathcal{M}$ is weakly ergodic if and only if there is an integer $r$ and real number $\alpha < 1$, such that $\|P_w\mu - P_w\nu\|_1 \leq \alpha$ for any word $w$ of length $r$. Our most general characterization of weak ergodicity is as follows: [11]:

**Theorem 1** *An abstract MCS $\mathcal{M}$ is weakly ergodic if and only if there exists a multiplicative operator's norm $\| \cdot \|_{**}$ on $\mathcal{L}$ equivalent to the norm $\| \cdot \|_{\mathcal{B}} :=$ $\sup_{\{\lambda:\lambda\Omega=0\}} \frac{\|P\lambda\|_1}{\|\lambda\|_1}$, and such that $\sup_{u\in\Sigma}\|P_u\|_{**} \leq \varepsilon$ for some number $\varepsilon < 1$.* ∎

The next theorem connects the computational power of weakly ergodic MCS's with the class of definite languages, generalizing the results by Rabin [9], Paz [8, p. 175], and Maass and Sontag [5].

**Theorem 2** *Let $\mathcal{M}$ be a weakly ergodic MCS. If a language $L$ can be recognized by $\mathcal{M}$, then it is definite.* ∎

## 4   The Stability Theorem of Weakly Ergodic MCS

An important issue for any computational system is whether the machine is robust with respect to small perturbations of the system's parameters or under some external noise. The stability of language recognition by weakly ergodic MCS's under perturbations of their Markov operators was previously considered by Rabin [9] and Paz [7, 8]. We next state a general version of the stability theorem that is applicable to our wide notion of weakly ergodic systems.

We first define two MCS's $\mathcal{M}$ and $\widetilde{\mathcal{M}}$ to be *similar* if they share the same measurable space $(\Omega, \mathcal{B})$, alphabet $\Sigma$, and sets $\mathcal{A}$ and $\mathcal{R}$, and if they differ only by their associated Markov operators.

**Theorem 3** *Let $\mathcal{M}$ and $\widetilde{\mathcal{M}}$ be two similar MCS's such that the first is weakly ergodic. Then there is $\alpha > 0$, such that if $\|P_u - \tilde{P}_u\|_1 \leq \alpha$ for all $u \in \Sigma$, then the second is also weakly ergodic. Moreover, these two MCS's recognize exactly the same class of languages.* ∎

**Corollary 3.1** *Let $\mathcal{M}$ and $\widetilde{\mathcal{M}}$ be two similar MCS's. Suppose that the first is weakly ergodic. Then there exists $\beta > 0$, such that if $\sup_{A\in\mathcal{B}} |P_u(x, A) - \tilde{P}_u(x, A)| \leq \beta$ for all $u \in \Sigma, x \in \Omega$, the second is also weakly ergodic. Moreover, these two MCS's recognize exactly the same class of languages.* ∎

A mathematically deeper result which implies Theorem 3 was proven in [11]:

**Theorem 4** *Let $\mathcal{M}$ and $\widetilde{\mathcal{M}}$ be two similar MCS's, such that the first is weakly ergodic and the second is arbitrary. Then, for any $\alpha > 0$ there exists $\varepsilon > 0$ such that $\|P_u - \tilde{P}_u\|_1 \leq \varepsilon$ for all $u \in \Sigma$ implies $\|P_w - \tilde{P}_w\|_1 \leq \alpha$ for all words $w \in \Sigma^*$.* ∎

Theorem 3 follows from Theorem 4. To see this, one can chose any $\alpha < \rho$ in Theorem 4 and observe that $\|P_w - \tilde{P}_w\|_1 \leq \alpha < \rho$ implies that the word $w$ is accepted or rejected by $\widetilde{\mathcal{M}}$ in accordance to whether it is accepted or rejected by $\mathcal{M}$.

## 5   Conditions on the Transition Probabilities

This section discusses practical conditions for weakly ergodic MCS's in which the Markov operators $P_u$ are induced by transition probability functions as in (4). Clearly, a simple sufficient condition for an MCS to be weakly ergodic is given by $\sup_{u \in \Sigma} \delta(P_u) \leq 1 - c$, for some $c > 0$.

Maass and Sontag used Doeblin's condition to prove the computational power of noisy neural networks [5]. Although the networks in [5] constitute a very particular case of weakly ergodic MCS's, Doeblin's condition is applicable also to our general model. The following version of Doeblin's condition was given by Doob [3]:

**Definition 5.1** [3] Let $P(x, A)$ be a TPF on $(\Omega, \mathcal{B})$. We say that it satisfies Doeblin condition, $D_0^n$, if there exists a constant $c$ and a probability measure $\mu$ on $(\Omega, \mathcal{B})$ such that $P^n(x, A) \geq c\mu(A)$ for any set $A \in \mathcal{B}$. ∎

If an MCS $\mathcal{M}$ is weakly ergodic, then all its associated TPF $P_w(x, A), w \in \Sigma$ must satisfy $D_0^n$ for some $n = n(w)$. Doob has proved [3, p. 197] that if $P(x, A)$ satisfies Doeblin's condition $D_0^1$ with constant c, then for any $\mu, \nu \in \mathcal{P}$, $\|P\mu - P\nu\|_1 \leq (1 - c)\|\mu - \nu\|_1$, i.e., $\delta(P) \leq 1 - c$. This leads us to the following definition.

**Definition 5.2** Let $\mathcal{M}$ be an MCS. We say that the space $\Omega$ is *small* with respect to $\mathcal{M}$ if there exists an $m > 0$ such that all associated TPF $P_w(x, A)$, $w \in \Sigma^m$ satisfy Doeblin's condition $D_0^1$ uniformly with the same constant $c$, i.e., $P_w(x, A) \geq c\mu_w(A)$, $w \in \Sigma^m$. ∎

The following theorem strengthens the result by Maass and Sontag [5].

**Theorem 5** *Let $\mathcal{M}$ be an MCS. If the space $\Omega$ is small with respect to $\mathcal{M}$, then $\mathcal{M}$ is weakly ergodic, and it can recognize only definite languages.* ∎

This theorem provides a convenient method for checking weak ergodicity in a given TPF. The theorem implies that it is sufficient to execute the following simple check: choose any integer $n$, and then verify that for every state $x$ and all input strings $w \in \Sigma^n$, the "absolutely continuous" part of all TPF $P_w, w \in \Sigma^n$ is uniformly bounded from below:

$$\psi_w\left(\{y : p_w(x, y) \geq c_1 \quad \text{for all } w \in \Sigma^n\right) \geq c_2, \tag{6}$$

where $p_w(x, y)$ is the density of the absolutely continuous component of $P_w(x, \cdot)$ with respect to $\psi_w$, and $c_1, c_2$ are positive numbers.

Most practical systems can be defined by null preserving TPF (including for example the systems in [5]). For these systems we provide (Theorem 6) a sufficient and necessary condition in terms of density kernels. A TPF $P_u(x, A)$, $u \in \Sigma$ is called *null preserving* with respect to a probability measure $\mu \in \mathcal{P}$ if it has a density with respect to $\mu$ i.e., $P(x, A) = \int_A p_u(x, z)\mu(dz)$. It is not hard to see, that the property of null preserving per letter $u \in \Sigma$ implies that all TPF $P_w(x, A)$ of words $w \in \Sigma^*$ are null preserving as well. In this case $\delta(P_u) = 1 - \inf_{x,y} \int_\Omega \min\{p_u(x, z), p_u(y, z)\}\mu_u(dz)$ and we have:

**Theorem 6** *Let $\mathcal{M}$ be an MCS defined by null preserving transition probability functions $P_u, u \in \Sigma$. Then, $\mathcal{M}$ is weakly ergodic if and only if there exists n such that $\inf_{w \in \Sigma^n} \inf_{x,y} \int_\Omega \min\{p_u(x, z), p_u(y, z)\}\mu_u(dz) > 0$.* ∎

A similar result was previously established by Paz [7, 8] for the case of a denumerable state space $\Omega$. This theorem allows to treat examples which are not covered by

Theorem 5. For example, suppose that the space $\Omega$ is not small with respect to an MCS $\mathcal{M}$, but for some $n$ and any $w \in \Sigma^n$ there exists a measure $\psi_w$ on $(\Omega, \mathcal{B})$ with the property that for any couple of states $x, y \in \Omega$

$$\psi_w \left( \{ z : \min\{p_w(x,z), p_w(y,z)\} \geq c_1 \} \right) \geq c_2, \tag{7}$$

where $p_w(x,y)$ is the density of $P_w(x, \cdot)$ with respect to $\psi_w$, and $c_1, c_2$ are positive numbers. This condition may occur even if there is no $y$ such that $p_u(x,y) \leq c_1$ for all $x \in \Omega$.

## 6   Examples of Weakly Ergodic Systems

### 1. The Synchronous Parallel Model

Let $(\Omega_i, \mathcal{B}_i), i = 1, 2, ..., N$ be a collection of measurable sets. Define $\Omega^i = \prod_{j \neq i} \Omega_j$ and $\mathcal{B}^i = \prod_{j \neq i} \mathcal{B}_j$. Then $(\Omega^i, \mathcal{B}^i)$ are measurable spaces. Define also $\Sigma_i = \Sigma \times \Omega^i$, and $T_i = \{ P_{\mathbf{x}^i, u}(x_i, A_i) : (\mathbf{x}^i, u) \in \Sigma_i \}$ be given stochastic kernels. Each set $T_i$ defines an MCS $\mathcal{M}_i$. We can define an *aggregate* MCS by setting $\Omega = \prod_i \Omega_i$, $\mathcal{B} = \prod_i \mathcal{B}_i$, $S = \prod_i S_i$, $R = \prod_i R_i$, and

$$P_u(x, A) = \prod_i P_{\mathbf{x}^i, u}(x_i, A_i). \tag{8}$$

This describes a model of $N$ noisy computational systems that update in synchronous parallelism. The state of the whole aggregate is a vector of states of the individual components, and each receives the states of all other components as part of its input.

**Theorem 7** *[12] Let $\mathcal{M}$ be an MCS defined by equation (8). It is weakly ergodic if at least one set of operators $T^i$ is such that $\delta(P^i_{u, \mathbf{x}^i}) \leq 1 - c$ for any $u \in \Sigma$, $\mathbf{x}^i \in \Omega^i$ and some positive number $c$.*                                                                    ■

### 2. The Asynchronous Parallel Model

In this model, at every step only one component is activated. Suppose that a collection of N *similar* MCS's $\mathcal{M}_i, i = 1, ..., N$ is given. Consider a probability measure $\varepsilon = \{\varepsilon_1, ..., \varepsilon_N\}$ on the set $K = \{1, ..., N\}$. Assume that in each computational step only one MCS is activated. The current state of the whole aggregate is represented by the state of its active component. Assume also that the probability of a computational system $\mathcal{M}_i$ to be activated, is time-independent and is given by $Prob(\mathcal{M}_i) = \varepsilon_i$. The aggregate system is then described by stochastic kernels

$$P_u(x, A) = \sum_{i=1}^{N} \varepsilon_i P^i_u(x, A). \tag{9}$$

**Theorem 8** *[12] Let $\mathcal{M}$ be an MCS defined by formula (9). It is weakly ergodic if at least one set of operators $\{P^1_u\}, ..., \{P^N_u\}$ is weakly ergodic.*                              ■

### 3. Hybrid Weakly Ergodic Systems

We now present a hybrid weakly ergodic computational system consisting of both continuous and discrete elements. The evolution of the system is governed by a differential equation, while its input arrives at discrete times. Let $\Omega = \mathbb{R}^n$, and consider a collection of differential equations

$$\dot{x}_u(s) = \psi_u(x_u(s)), \ u \in \Sigma, \ s \in [0, \infty). \tag{10}$$

Suppose that $\psi_u(x)$ is sufficiently smooth to ensure the existence and uniqueness of solutions of Equation (10) for $s \in [0, 1]$ and for any initial condition.

Consider a computational system which receives an input $u(t)$ at discrete times $t_0, t_1, t_2 \ldots$. In the interval $t \in [t_i, t_{i+1}]$ the behavior of the system is described by Equation (10), where $s = t - t_i$. A random initial condition for the time $t_n$ is defined by

$$Prob[x_{u(t_n)}(0) \in A] = P_u(x_{u(t_{n-1})}(1), A), \tag{11}$$

where $x_{u(t_{n-1})}(1)$ is the state of the system after previously completed computations, and $P_u(x, A)$, $u \in \Sigma$ is a family of stochastic kernels on $\Omega \times B$. This describes a system which receives inputs in discrete instants of time; the input letters $u \in \Sigma$ cause random perturbations of the state $x_{u(t-1)}(1)$ governed by the transition probability functions $P_{u(t)}(x_{u(t-1)}, A)$. In all other times the system is a noise-free continuous computational system which evolves according to equation (10).

Let $\Omega = \mathbb{R}^n$, $x_0 \in \Omega$ be a distinguished initial state, and let $S$ and $R$ be two subsets of $\Omega$ with the property of having a $\rho$-gap: $dist(S, R) = \inf_{x \in S, y \in R} \|x - y\| = \rho > 0$. The first set is called a set of *accepting final states* and the second is called a set of *rejecting final states*. We say that the *hybrid computational system* $\mathcal{M} = \langle \Omega, \Sigma, x_0, \psi_u, S, R \rangle$ recognizes $L \subseteq \Sigma^*$ if for all $w = w_0 \ldots w_n \in \Sigma^*$ and the end letter $\$ \notin \Sigma$ the following holds: $w \in L \Leftrightarrow Prob(x_{w_n\$}(1) \in S) > \frac{1}{2} + \varepsilon$, and $w \notin L \Leftrightarrow Prob(x_{w_n\$}(1) \in R) > \frac{1}{2} + \varepsilon$.

**Theorem 9** *[12] Let $\mathcal{M}$ be a hybrid computational system. It is weakly ergodic if its set of evolution operators $T = \{P_u : u \in \Sigma\}$ is weakly ergodic.*  ∎

# References

[1] Casey, M., The Dynamics of Discrete-Time Computation, With Application to Recurrent Neural Networks and Finite State Machine Extraction, Neural Computation 8, 1135-1178, 1996.

[2] Dobrushin, R. L., Central limit theorem for nonstationary Markov chains I, II. *Theor. Probability Appl.* vol. 1, 1956, pp 65–80, 298–383.

[3] Doob J. L., *Stochastic Processes.* John Wiley and Sons, Inc., 1953.

[4] W. Maass and Orponen, P., On the effect of analog noise in discrete time computation, *Neural Computation*, 10(5), 1998, pp. 1071–1095.

[5] W. Maass and Sontag, E., Analog neural nets with Gaussian or other common noise distribution cannot recognize arbitrary regular languages, *Neural Computation*, 11, 1999, pp. 771–782.

[6] Neveu J., *Mathematical Foundations of the Calculus of Probability.* Holden Day, San Francisco, 1964.

[7] Paz A., Ergodic theorems for infinite probabilistic tables. *Ann. Math. Statist.* vol. 41, 1970, pp. 539–550.

[8] Paz A., *Introduction to Probabilistic Automata. Academic Press*, Inc., London, 1971.

[9] Rabin, M., Probabilistic automata, *Information and Control*, vol 6, 1963, pp. 230-245.

[10] Siegelmann H. T., *Neural Networks and Analog Computation: Beyond the Turing Limit.* Birkhauser, Boston, 1999.

[11] Siegelmann H. T. and Roitershtein A., On weakly ergodic computational systems, 1999, *submitted.*

[12] Siegelmann H. T., Roitershtein A., and Ben-Hur, A., On noisy computational systems, 1999, *Discrete Applied Mathematics*, accepted.
